# Temporal Difference Learning of Position Evaluation in the Game of Go

**Nicol N. Schraudolph**
schraudo@salk.edu

**Peter Dayan**
dayan@salk.edu

**Terrence J. Sejnowski**
terry@salk.edu

Computational Neurobiology Laboratory
The Salk Institute for Biological Studies
San Diego, CA 92186-5800

## Abstract

The game of Go has a high branching factor that defeats the tree search approach used in computer chess, and long-range spatiotemporal interactions that make position evaluation extremely difficult. Development of conventional Go programs is hampered by their knowledge-intensive nature. We demonstrate a viable alternative by training networks to evaluate Go positions via temporal difference (TD) learning.

Our approach is based on network architectures that reflect the spatial organization of both input and reinforcement signals on the Go board, and training protocols that provide exposure to competent (though unlabelled) play. These techniques yield far better performance than undifferentiated networks trained by self-play alone. A network with less than 500 weights learned within 3,000 games of 9x9 Go a position evaluation function that enables a primitive one-ply search to defeat a commercial Go program at a low playing level.

## 1 INTRODUCTION

Go was developed three to four millenia ago in China; it is the oldest and one of the most popular board games in the world. Like chess, it is a deterministic, perfect information, zero-sum game of strategy between two players. They alternate in

placing black and white stones on the intersections of a 19x19 grid (smaller for beginners) with the objective of surrounding more board area *(territory)* with their stones than the opponent. Adjacent stones of the same color form *groups;* an empty intersection adjacent to a group is called a *liberty* of that group. A group is captured and removed from the board when its last liberty is occupied by the opponent. To prevent loops, it is illegal to make a move which recreates a prior board position. A player may pass at any time; the game ends when both players pass in succession.

Unlike most other games of strategy, Go has remained an elusive skill for computers to acquire — indeed it has been recognized as a "grand challenge" of Artificial Intelligence (Rivest, 1993). The game tree search approach used extensively in computer chess is infeasible: the game tree of Go has an average branching factor of around 200, but even beginners may routinely look ahead up to 60 plies in some situations. Humans appear to rely mostly on static evaluation of board positions, aided by highly selective yet deep local lookahead. Conventional Go programs are carefully (and protractedly) tuned expert systems (Fotland, 1993). They are fundamentally limited by their need for human assistance in compiling and integrating domain knowledge, and still play barely above the level of a human beginner — a machine learning approach may thus offer considerable advantages. (Brügmann, 1993) has shown that a knowledge-free optimization approach to Go can work in principle: he obtained respectable (though inefficient) play by selecting moves through simulated annealing (Kirkpatrick et al., 1983) over possible continuations of the game.

The pattern recognition component inherent in Go is amenable to connectionist methods. Supervised backpropagation networks have been applied to the game (Stoutamire, 1991; Enderton, 1991) but face a bottleneck in the scarcity of hand-labelled training data. We propose an alternative approach based on the TD($\lambda$) predictive learning algorithm (Sutton, 1984; Sutton, 1988; Barto et al., 1983), which has been successfully applied to the game of backgammon by (Tesauro, 1992). His TD-Gammon program uses a backpropagation network to map preselected features of the board position to an output reflecting the probability that the player to move would win. It was trained by TD(0) while playing only itself, yet learned an evaluation function that — coupled with a full two-ply lookahead to pick the estimated best move — made it competitive with the best human players in the world (Robertie, 1992; Tesauro, 1994).

In an early experiment we investigated a straightforward adaptation of Tesauro's approach to the Go domain. We trained a fully connected 82-40-1 backpropagation network by randomized[1] self-play on a 9x9 Go board (a standard didactic size for humans). The output learned to predict the margin of victory or defeat for black. This undifferentiated network did learn to squeak past *Wally,* a weak public domain program (Newman, 1988), but it took 659,000 games of training to do so. We have found that the efficiency of learning can be vastly improved through appropriately structured network architectures and training strategies, and these are the focus of the next two sections.

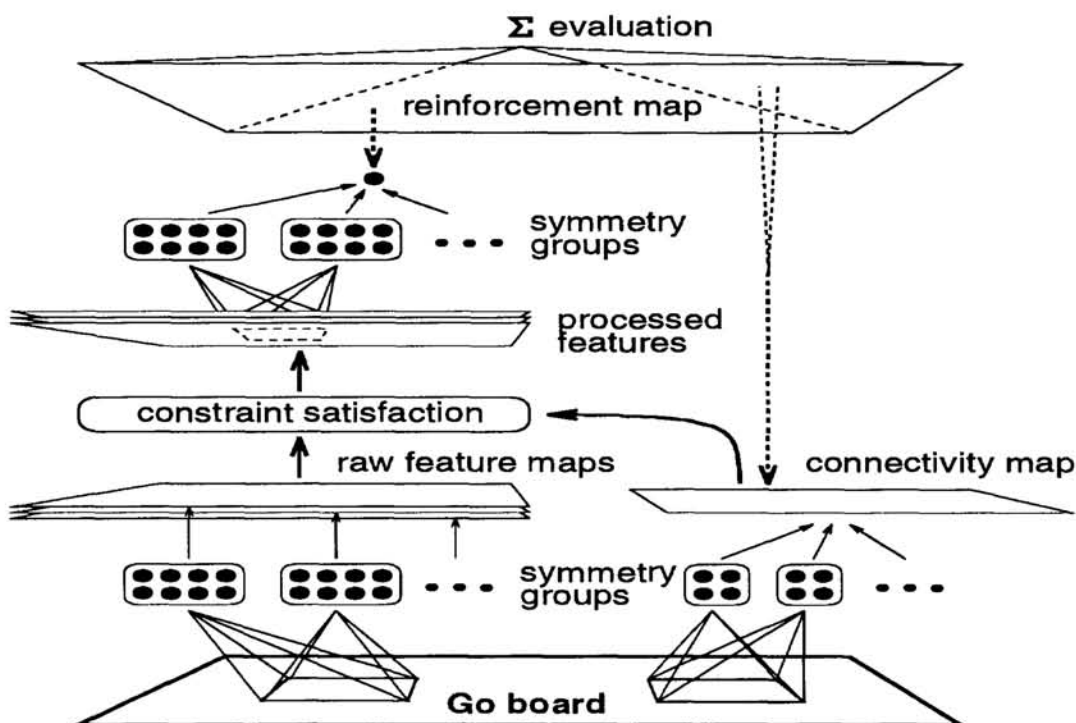

Figure 1: A modular network architecture that takes advantage of board symmetries, translation invariance and localized reinforcement to evaluate Go positions. Also shown is the planned connectivity prediction mechanism (see Discussion).

## 2  NETWORK ARCHITECTURE

One of the particular advantages of Go for predictive learning is that there is much richer information available at the end of the game than just who won. Unlike chess, checkers or backgammon, in which pieces are taken away from the board until there are few or none left, Go stones generally remain where they are placed. This makes the final state of the board richly informative with respect to the course of play; indeed the game is scored by summing contributions from each point on the board. We make this spatial credit assignment accessible to the network by having it predict the fate of every point on the board rather than just the overall score, and evaluate whole positions accordingly. This bears some similarity with the Successor Representation (Dayan, 1993) which also integrates over vector rather than scalar destinies.[2]

Given the knowledge-based approach of existing Go programs, there is an embarrassment of input features that one might adopt for Go: *Wally* already uses about 30 of them, stronger programs disproportionately more. In order to demonstrate reinforcement learning as a viable alternative to the conventional approach, however, we require our networks to learn whatever set of features they might need. The complexity of this task can be significantly reduced by exploiting a number

of constraints that hold *a priori* in this domain. Specifically, patterns of Go stones retain their properties under color reversal, reflection and rotation of the board, and — modulo the considerable influence of the board edges — translation. Each of these invariances is reflected in our network architecture:

Color reversal invariance implies that changing the color of every stone in a Go position, and the player whose turn it is to move, yields an equivalent position from the other player's perspective. We build this constraint directly into our networks by using antisymmetric input values (+1 for black, -1 for white) and squashing functions throughout, and negating the bias input when it is white's turn to move.

Go positions are also invariant with respect to the eightfold (reflection × rotation) symmetry of the square. We provided mechanisms for constraining the network to obey this invariance by appropriate weight sharing and summing of derivatives (Le Cun et al., 1989). Although this is clearly beneficial during the evaluation of the network against its opponents, it appears to impede the course of learning.[3]

To account for translation invariance we use convolution with a weight kernel rather than multiplication by a weight matrix as the basic mapping operation in our network, whose layers are thus *feature maps* produced by scanning a fixed receptive field across the input. One particular advantage of this technique is the easy transfer of learned weight kernels to different Go board sizes.

It must be noted, however, that Go is *not* translation-invariant: the edge of the board not only affects local play but modulates other aspects of the game, and indeed forms the basis of opening strategy. We currently account for this by allowing each node in our network to have its own bias weight, giving it one degree of freedom from its neighbors. This enables the network to encode absolute position at a modest increse in the number of adjustable parameters. Furthermore, we provide additional redundancy around the board edges by selective use of convolution kernels twice as wide as the input.

Figure 1 illustrates the modular architecture suggested by these deliberations. In the experiments described below we implement all the features shown except for the connectivity map and lateral constraint satisfaction, which are the subject of future work.

## 3   TRAINING STRATEGIES

Temporal difference learning teaches the network to predict the consequences of following particular strategies on the basis of the play they produce. The question arises as to which strategies should be used to generate the large number of Go games needed for training. We have identified three criteria by which we compare alternative training strategies:

- the computational **efficiency** of move generation,
- the **quality** of generated play, and
- reasonable **coverage** of plausible Go positions.

Tesauro trained TD-Gammon by self-play — *ie.* the network's own position evaluation was used in training to pick both players' moves. This technique does not require any external source of expertise beyond the rules of the game: the network is its own teacher. Since Go is a deterministic game, we cannot always pick the estimated best move when training by self-play without running the risk of trapping the network in some suboptimal fixed state. Theoretically, this should not happen — the network playing white would be able to predict the idiosyncrasies of the network playing black, take advantage of them thus changing the outcome, and forcing black's predictions to change commensurately — but in practice it is a concern. We therefore pick moves stochastically by Gibbs sampling (Geman and Geman, 1984), in which the probability of a given move is exponentially related to the predicted value of the position it leads to through a "temperature" parameter that controls the degree of randomness.

We found self-play alone to be rather cumbersome for two reasons: firstly, the single-ply search used to evaluate all legal moves is computationally intensive — and although we are investigating faster ways to accomplish it, we expect move evaluation to remain a computational burden. Secondly, learning from self-play is sluggish as the network must bootstrap itself out of ignorance without the benefit of exposure to skilled opponents. However, there is nothing to keep us from training the network on moves that are not based on its own predictions — for instance, it can learn by playing against a conventional Go program, or even by just observing games between human players.

We use three computer opponents to train our networks: a random move generator, the public-domain program *Wally* (Newman, 1988), and the commercial program *The Many Faces of Go* (Fotland, 1993). The random move generator naturally doesn't play Go very well[4], but it does have the advantages of high speed and ergodicity — a few thousand games of random Go proved an effective way to prime our networks at the start of training. The two conventional Go programs, by contrast, are rather slow and deterministic, and thus not suitable generators of training data when playing among themselves. However, they do make good opponents for the network, which can provide the required variety of play through its Gibbs sampler. When training on games played between such dissimilar players, we must match their strength so as to prevent trivial predictions of the outcome. Against *Many Faces* we use standard Go handicaps for this purpose; *Wally* we modified to intersperse its play with random moves. The proportion of random moves is reduced adaptively as the network improves, providing us with an on-line performance measure.

Since, in all cases, the strategies of both players are intimately intertwined in the predictions, one would never expect them to be correct overall when the network is playing a real opponent. This is a particular problem when the strategy for choosing moves during learning is different from the policy adopted for 'optimal' network play. (Samuel, 1959) found it inadvisable to let his checker program learn from games which it won against an opponent, since its predictions might otherwise reflect poor as well as good play. This is a particularly pernicious form of over-fitting — the network can learn to predict one strategy in exquisite detail, without being able to play well in general.

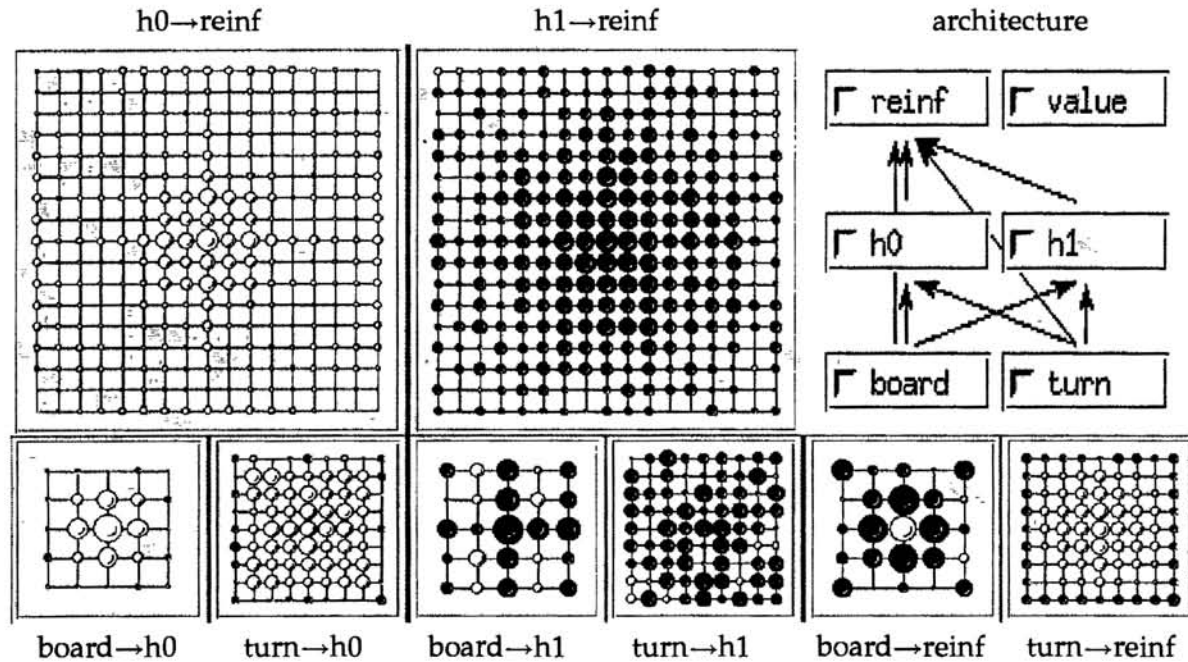

Figure 2: A small network that learned to play 9x9 Go. Boxes in the architecture panel represent 9x9 layers of units, except for *turn* which is a single bias unit. Arrows indicate convolutions with the corresponding weight kernels. Black disks represent excitatory, white ones inhibitory weights; within each matrix, disk area is proportional to weight magnitude.

## 4   RESULTS

In exploring this domain, we trained many networks by a variety of methods. A small sample network that learned to beat *Many Faces* (at low playing level) in 9x9 Go within 3,000 games of training is shown in Figure 2. This network was grown during training by adding hidden layers one at a time; although it was trained without the (reflection × rotation) symmetry constraint, many of the weight kernels learned approximately symmetric features. The direct projection from board to reinforcement layer has an interesting structure: the negative central weight within a positive surround stems from the fact that a placed stone occupies (thus loses) a point of territory even while securing nearby areas. Note that the wide 17x17 projections from the hidden layers have considerable fringes — ostensibly a trick the network uses to incorporate edge effects, which are also prominent in the bias projections from the *turn* unit.

We compared training this architecture by self-play versus play against *Wally*. The initial rate of learning is similar, but soon the latter starts to outperform the former (measured against both *Wally* and *Many Faces*), demonstrating the advantage of having a skilled opponent. After about 2000 games, however, it starts to overfit to *Wally* and consequently worsens against *Many Faces*. Switching training partner to *Many Faces* at this point produced (after a further 1,000 games) a network that could reliably beat this opponent. Although less capable, the self-play network did manage to edge past *Wally* after 3,000 games; this compares very favorably with

the undifferentiated network described in the Introduction. Furthermore, we have verified that weights learned from 9x9 Go offer a suitable basis for further training on the full-size (19x19) board.

# 5   DISCUSSION

In general our networks appear more competent in the opening than further into the game. This suggests that although reinforcement information is indeed propagating all the way back from the final position, it is hard for the network to capture the multiplicity of mid-game situations and the complex combinatorics characteristic of the endgame. These strengths and weaknesses partially complement those of symbolic systems, suggesting that hybrid approaches might be rewarding. We plan to further improve network performance in a number of ways:

It is possible to augment the input representation of the network in such a way that its task becomes fully translation-invariant. We intend to do this by adding an extra input layer whose nodes are active when the corresponding points on the Go board are empty, and inactive when they are occupied (regardless of color). Such an explicit representation of *liberties* makes the three possible states of a point on the board (black stone, white stone, or empty) linearly separable to the network, and eliminates the need for special treatment of the board edges.

The use of limited receptive field sizes raises the problem of how to account for long-ranging spatial interactions on the board. In Go, the distance at which groups of stones interact is a function of their arrangement in context; an important subproblem of position evaluation is therefore to compute the *connectivity* of groups of stones. We intend to model connectivity explicitly by training the network to predict the *correlation* pattern of local reinforcement from a given position. This information can then be used to control the lateral propagation of local features in the hidden layer through a constraint satisfaction mechanism.

Finally, we can train networks on recorded games between human players, which the *Internet Go Server* provides in steady quantities and machine-readable format. We are only beginning to explore this promising supply of instantaneous (since prerecorded), high-quality Go play for training. The main obstacle encountered so far has been the human practice of abandoning the game once both players agree on the outcome — typically well before a position that could be scored mechanically is reached. We address this issue by eliminating early resignations from our training set, and using *Wally* to bring the remaining games to completion.

We have shown that with sufficient attention to network architecture and training procedures, a connectionist system trained by temporal difference learning alone can achieve significant levels of performance in this knowledge-intensive domain.

**Acknowledgements**

We are grateful to Patrice Simard and Gerry Tesauro for helpful discussions, to Tim Casey for the plethora of game records from the Internet Go Server, and to Geoff Hinton for *tniterations*. Support was provided by the McDonnell-Pew Center for Cognitive Neuroscience, SERC, NSERC and the Howard Hughes Medical Institute.

**References**

Barto, A., Sutton, R., and Anderson, C. (1983). Neuronlike adaptive elements that can solve difficult learning control problems. *IEEE Transactions on Systems, Man, and Cybernetics*, 13.

Brügmann, B. (1993). Monte Carlo Go. Manuscript available by Internet anonymous file transfer from bsdserver.ucsf.edu, file Go/comp/mcgo.tex.Z.

Dayan, P. (1993). Improving generalization for temporal difference learning: The successor representation. *Neural Computation*, 5(4):613–624.

Enderton, H. D. (1991). The Golem Go program. Technical Report CMU-CS-92-101, Carnegie Mellon University. Report available by Internet anonymous file transfer from bsdserver.ucsf.edu, file Go/comp/golem.sh.Z.

Fotland, D. (1993). Knowledge representation in the Many Faces of Go. Manuscript available by Internet anonymous file transfer from bsdserver.ucsf.edu, file Go/comp/mfg.Z.

Geman, S. and Geman, D. (1984). Stochastic relaxation, gibbs distributions, and the bayesian restoration of images. *IEEE Transactions on Pattern Analysis and Machine Intelligence*, 6.

Kirkpatrick, S., Gelatt Jr., C. D., and Vecchi, M. P. (1983). Optimization by simulated annealing. *Science*, 220:671–680.

Le Cun, Y., Boser, B., Denker, J., Henderson, D., Howard, R., Hubbard, W., and Jackel, L. (1989). Backpropagation applied to handwritten zip code recognition. *Neural Computation*, 1:541–551.

Newman, W. H. (1988). Wally, a Go playing program. Shareware C program available by Internet anonymous file transfer from bsdserver.ucsf.edu, file Go/comp/wally.sh.Z.

Rivest, R. (1993). MIT Press, forthcoming. Invited talk: Computational Learning Theory and Natural Learning Systems, Provincetown, MA.

Robertie, B. (1992). Carbon versus silicon: Matching wits with TD-Gammon. *Inside Backgammon*, 2(2):14–22.

Samuel, A. L. (1959). Some studies in machine learning using the game of checkers. *IBM Journal of Research and Development*, 3:211–229.

Stoutamire, D. (1991). Machine learning applied to Go. Master's thesis, Case Western Reserve University. Reprint available by Internet anonymous file transfer from bsdserver.ucsf.edu, file Go/comp/report.ps.Z.

Sutton, R. (1984). *Temporal Credit Assignment in Reinforcement Learning*. PhD thesis, University of Massachusetts, Amherst.

Sutton, R. (1988). Learning to predict by the methods of temporal differences. *Machine Learning*, 3:9–44.

Tesauro, G. (1992). Practical issues in temporal difference learning. *Machine Learning*, 8:257–278.

Tesauro, G. (1994). TD-Gammon, a self-teaching backgammon program, achieves master-level play. *Neural Computation*, 6(2):215–219.

## Footnotes

[1]Unlike backgammon, Go is a deterministic game, so we had to generate moves stochastically to ensure sufficient exploration of the state space. This was done by Gibbs sampling (Geman and Geman, 1984) over values obtained from single-ply search, annealing the temperature parameter from random towards best-predicted play.

[2]Sharing information within the network across multiple outputs restricts us to $\lambda = 0$ for efficient implementation of TD($\lambda$). Note that although (Tesauro, 1992) did not have this constraint, he nevertheless found $\lambda = 0$ to be optimal.

[3] We are investigating possible causes and cures for this phenomenon.

[4] In order to ensure a minimum of stability in the endgame, it does refuse to fill in its own *eyes* — a particular, locally recognizable type of suicidal move.
